# Learning Monotonic Transformations for Classification

**Andrew G. Howard**
Department of Computer Science
Columbia University
New York, NY 10027
ahoward@cs.columbia.edu

**Tony Jebara**
Department of Computer Science
Columbia University
New York, NY 10027
jebara@cs.columbia.edu

## Abstract

A discriminative method is proposed for learning monotonic transformations of the training data while jointly estimating a large-margin classifier. In many domains such as document classification, image histogram classification and gene microarray experiments, fixed monotonic transformations can be useful as a preprocessing step. However, most classifiers only explore these transformations through manual trial and error or via prior domain knowledge. The proposed method learns monotonic transformations automatically while training a large-margin classifier without any prior knowledge of the domain. A monotonic piecewise linear function is learned which transforms data for subsequent processing by a linear hyperplane classifier. Two algorithmic implementations of the method are formalized. The first solves a convergent alternating sequence of quadratic and linear programs until it obtains a locally optimal solution. An improved algorithm is then derived using a convex semidefinite relaxation that overcomes initialization issues in the greedy optimization problem. The effectiveness of these learned transformations on synthetic problems, text data and image data is demonstrated.

## 1 Introduction

Many fields have developed heuristic methods for preprocessing data to improve performance. This often takes the form of applying a monotonic transformation prior to using a classification algorithm. For example, when the bag of words representation is used in document classification, it is common to take the square root of the term frequency [6, 5]. Monotonic transforms are also used when classifying image histograms. In [3], transformations of the form $x^a$ where $0 \leq a \leq 1$ are demonstrated to improve performance. When classifying genes from various microarray experiments it is common to take the logarithm of the gene expression ratio [2]. Monotonic transformations can also capture crucial properties of the data such as threshold and saturation effects.

In this paper, we propose to simultaneously learn a hyperplane classifier and a monotonic transformation. The solution produced by our algorithm is a piecewise linear monotonic function and a maximum margin hyperplane classifier similar to a support vector machine (SVM) [4]. By allowing for a richer class of transforms learned at training time (as opposed to a rule of thumb applied during preprocessing), we improve classification accuracy. The learned transform is specifically tuned to the classification task. The main contributions of this paper include, a novel framework for estimating a monotonic transformation and a hyperplane classifier simultaneously at training time, an efficient method for finding a

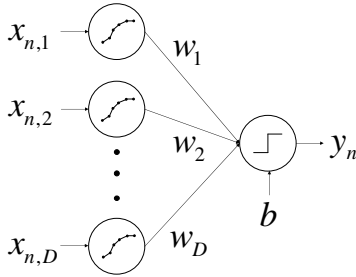

Figure 1: Monotonic transform applied to each dimension followed by a hyperplane classifier.

locally optimal solution to the problem, and a convex relaxation to find a globally optimal approximate solution.

The paper is organized as follows. In section 2, we present our formulation for learning a piecewise linear monotonic function and a hyperplane. We show how to learn this combined model through an iterative coordinate ascent optimization using interleaved quadratic and linear programs to find a local minimum. In section 3, we derive a convex relaxation based on Lasserre's method [8]. In section 4 synthetic experiments as well as document and image classification problems demonstrate the diverse utility of our method. We conclude with a discussion and future work.

## 2    Learning Monotonic Transformations

For an unknown distribution $P(\vec{x}, y)$ over inputs $\vec{x} \in \Re^d$ and labels $y \in \{-1, 1\}$, we assume that there is an unknown nuisance monotonic transformation $\Phi(x)$ and unknown hyperplane parameterized by $\vec{w}$ and $b$ such that predicting with $f(x) = \text{sign}(\vec{w}^T \Phi(\vec{x}) + b)$ yields a low expected test error $R = \int \frac{1}{2}|y - f(x)| dP(\vec{x}, y)$. We would like to recover $\Phi(\vec{x}), \vec{w}, b$ from a labeled training set $S = \{(\vec{x}_1, y_1), \ldots, (\vec{x}_N, y_N)\}$ which is sampled i.i.d. from $P(\vec{x}, y)$. The transformation acts elementwise as can be seen in Figure 1.

We propose to learn both a maximum margin hyperplane and the unknown transform $\Phi(x)$ simultaneously. In our formulation, $\Phi(x)$ is a piecewise linear function that we parameterize with a set of K knots $\{z_1, \ldots, z_K\}$ and associated positive weights $\{m_1, \ldots, m_K\}$ where $z_j \in \Re$ and $m_j \in \Re^+$. The transformation can be written as $\Phi(x) = \sum_{j=1}^{K} m_j \phi_j(x)$ where $\phi_j(x)$ are truncated ramp functions acting on vectors and matrices elementwise as follows:

$$\phi_j(x) = \begin{cases} 0 & x \leq z_j \\ \frac{x-z_j}{z_{j+1}-z_j} & z_j < x < z_{j+1} \\ 1 & z_{j+1} \leq x \end{cases} \tag{1}$$

This is a less common way to parameterize piecewise linear functions. The positivity constraints enforce monotonicity on $\Phi(x)$ for all $x$. A more common method is to parameterize the function value $\Phi(z)$ at each knot $z$ and apply order constraints between subsequent knots to enforce monotonicity. Values in between knots are found through linear interpolation. This is the method used in isotonic regression [10], but in practice, these are equivalent formulations. Using truncated ramp functions is preferable for numerous reasons. They can be easily precomputed and are sparse. Once precomputed, most calculations can be done via sparse matrix multiplications. The positivity constraints on the weights $\vec{m}$ will also yield a simpler formulation than order constraints and interpolation which becomes important in subsequent relaxation steps.

Figure 2a shows the truncated ramp function associated with knot $z_1$. Figure 2b shows a conic combination of truncated ramps that builds a piecewise linear monotonic function. Combining this with the support vector machine formulation leads us to the following learning problem:

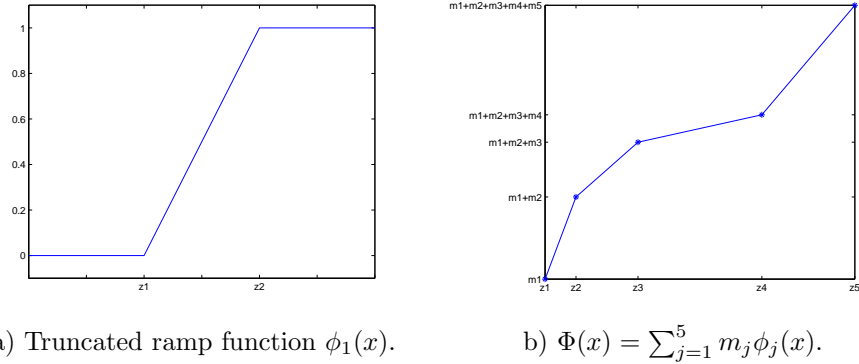

a) Truncated ramp function $\phi_1(x)$.          b) $\Phi(x) = \sum_{j=1}^{5} m_j \phi_j(x)$.

Figure 2: Building blocks for piecewise linear functions.

$$
\min_{\vec{w}, \vec{\xi}, b, \vec{m}} \quad \|\vec{w}\|_2^2 + C \sum_{i=1}^{N} \xi_i \tag{2}
$$

$$
\text{subject to} \quad y_i \left( \left\langle \vec{w}, \sum_{j=1}^{K} m_j \phi_j(\vec{x_i}) \right\rangle + b \right) \geq 1 - \xi_i \ \forall i
$$

$$
\xi_i \geq 0, m_j \geq 0, \sum_{j} m_j \leq 1 \ \forall i, j
$$

where $\vec{\xi}$ are the standard SVM slack variables, $\vec{w}$ and $b$ are the maximum margin solution for the training set that has been transformed via $\Phi(x)$ with learned weights $\vec{m}$. Before training, the knot locations are chosen at the empirical quantiles so that they are evenly spaced in the data.

This problem is nonconvex due to the quadratic term involving $\vec{w}$ and $\vec{m}$ in the classification constraints. Although it is difficult to find a globally optimal solution, the structure of the problem suggests a simple method for finding a locally optimal solution. We can divide the problem into two convex subproblems. This amounts to solving a support vector machine for $\vec{w}$ and $b$ with a fixed $\Phi(x)$ and alternatively solving for $\Phi(x)$ as a linear program with the SVM solution fixed. In both subproblems, we optimize over $\vec{\xi}$ as it is part of the hinge loss. This yields an efficient convergent optimization method. However, this method can get stuck in local minima. In practice, we initialize it with a linear $\Phi(x)$ and iterate from there. Alternative initializations do not yield much help. This leads us to look for a method to efficiently find global solutions.

## 3   Convex Relaxation

When faced with a nonconvex quadratic problem, an increasingly popular technique is to relax it into a convex one. Lasserre [8] proposed a sequence of convex relaxations for these types of nonconvex quadratic programs. This method replaces all quadratic terms in the original optimization problem with entries in a matrix. In its simplest form this matrix corresponds to the outer product of the the original variables with rank one and semidefinite constraints. The relaxation comes from dropping the rank one constraint on the outer product matrix. Lasserre proposed more elaborate relaxations using higher order moments of the variables. However, we mainly use the first moment relaxation along with a few of the second order moment constraints that do not require any additional variables beyond the outer product matrix.

A convex relaxation could be derived directly from the primal formulation of our problem. Both $\vec{w}$ and $\vec{m}$ would be relaxed as they interact in the nonconvex quadratic terms. Un-

fortunately, this yields a semidefinite constraint that scales with both the number of knots and the dimensionality of the data. This is troublesome because we wish to work with high dimensional data such as a bag of words representation for text. However, if we first find the dual formulation for $\vec{w}$, $b$, and $\vec{\xi}$, we only have to relax $\vec{m}$ which yields both a tighter relaxation and a less computationally intensive problem. Finding the dual leaves us with the following min max saddle point problem that will be subsequently relaxed and transformed into a semidefinite program:

$$\min_{\vec{m}} \max_{\vec{\alpha}} \quad 2\vec{\alpha}^T \vec{1} - \vec{\alpha}^T \left( Y \left( \sum_{i,j} m_i m_j \phi_i(X)^T \phi_j(X) \right) Y \right) \vec{\alpha} \qquad (3)$$

$$0 \le \alpha_i \le C, \vec{\alpha}^T \vec{y} = 0, m_j \ge 0, \sum_j m_j \le 1 \, \forall i,j$$

where $\vec{1}$ is a vector of ones, $\vec{y}$ is a vector of the labels, $Y = \mathrm{diag}(\vec{y})$ is a matrix with the labels on its diagonal with zeros elsewhere, and $X$ is a matrix with $\vec{x}_i$ in the $i$th column.

We introduce the relaxation via the substitution $M = \bar{m}\bar{m}^T$ and constraint $M \succeq 0$ where $\bar{m}$ is constructed by concatenating 1 with $\vec{m}$. We can then transform the relaxed min max problem into a semidefinite program similar to the multiple kernel learning framework [7] by finding the dual with respect to $\vec{\alpha}$ and using the Schur complement lemma to generate a linear matrix inequality [1]:

$$\min_{M,t,\lambda,\vec{\nu},\vec{\delta}} \quad t \qquad (4)$$

$$\text{subject to} \quad \begin{pmatrix} Y \sum_{i,j} M_{i,j} \phi_i(X)^T \phi_j(X) Y & \vec{1} + \vec{\nu} - \vec{\delta} + \lambda \vec{y} \\ (\vec{1} + \vec{\nu} - \vec{\delta} + \lambda \vec{y})^T & t - 2C\vec{\delta}^T \vec{1} \end{pmatrix} \succeq 0$$

$$M \succeq 0, M \ge 0, M\bar{1} \le \vec{0}, M_{0,0} = 1, \vec{\nu} \ge \vec{0}, \vec{\delta} \ge \vec{0}$$

where $\vec{0}$ is a vector of zeros and $\bar{1}$ is a vector with $-1$ in the first dimension and ones in the rest. The variables $\lambda$, $\vec{\nu}$, $\vec{\delta}$ arise from the dual transformation. This relaxation is exact if $M$ is a rank one matrix.

The above can be seen as a generalization of the multiple kernel learning framework. Instead of learning a kernel from a combination of kernels, we are learning a combination of inner products of different functions applied to our data. In our case, these are truncated ramp functions. The terms $\phi_i(X)^T \phi_j(X)$ are not Mercer kernels except when $i = j$. This more general combination requires the stricter constraints that the mixing weights $M$ form a positive semidefinite matrix, a constraint which is introduced via the relaxation. This is a sufficient condition for the resulting matrix $\sum_{i,j} M_{i,j} \phi_i(X)^T \phi_j(X)$ to also be positive semidefinite.

When using this relaxation, we can recover the monotonic transform by using the first column (row) as the mixing weights, $\vec{m}$, of the truncated ramp functions. In practice, however, we use the learned kernel in our predictions $k(\vec{x}, \vec{x}') = \sum_{i,j} M_{i,j} \phi_i(\vec{x})^T \phi_j(\vec{x}')$.

## 4 Experiments

### 4.1 Synthetic Experiment

In this experiment we will demonstrate our method's ability to recover a monotonic transformation from data. We sampled data near a linear decision boundary and generated labels based on this boundary. We then applied a strictly monotonic function to this sampled data. The training set is made up of the transformed points and the original labels. A linear algorithm will have difficulty because the mapped data is not linearly separable. However,

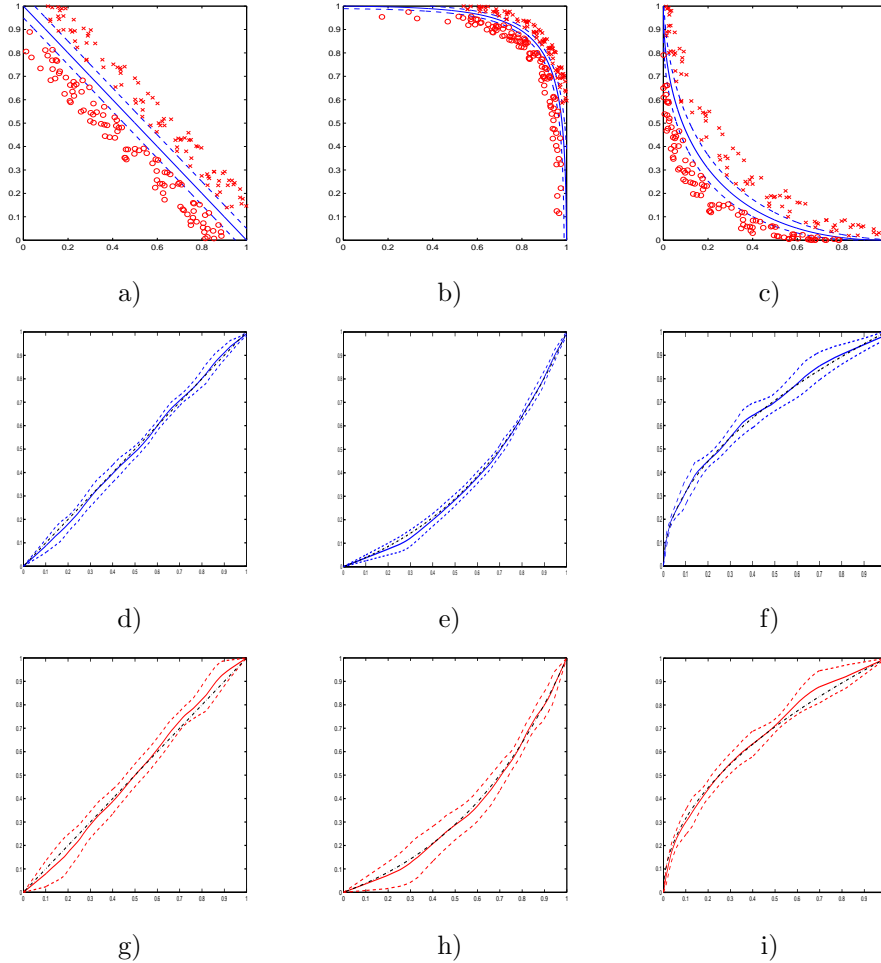

Figure 3: a) Original data. b) Data transformed by a logarithm. c) Data transformed by a quadratic function. d-f) The transformation functions learned using the nonconvex algorithm. g-i) The transformation functions learned using the convex algorithm.

if we could recover the inverse monotonic function, then a linear decision boundary would perform well.

Figure 3a shows the original data and decision boundary. Figure 3b shows the data and hyperplane transformed with a normalized logarithm. Figure 3c depicts a quadratic transform. 600 data points were sampled, and then transformed. 200 were used for training, 200 for cross validation and 200 for testing. We compared our locally optimal method (L mono), our convex relaxation (C mono) and a linear SVM (linear). The linear SVM struggled on all of the transformed data while the other methods performed well as reported in Figure 4. The learned transforms for L mono are plotted in Figure 3(d-f). The solid blue line is the mean over 10 experiments, and the dashed blue is the standard deviation. The black line is the true target function. The learned functions for C mono are in Figure 3(g-i). Both algorithms performed quite well on the task of classification and recover nearly the exact monotonic transform. The local method outperformed the relaxation slightly because this was an easy problem with few local minima.

## 4.2 Document Classification

In this experiment we used the four universities WebKB dataset. The data is made up of web pages from four universities plus an additional larger set from miscellaneous universities.

|         | linear     | exponential | square root | total  |
|---------|------------|-------------|-------------|--------|
| Linear  | **0.0005** | 0.0375      | 0.0685      | 0.0355 |
| L Mono  | 0.0020     | **0.0005**  | **0.0020**  | **0.0015** |
| C Mono  | 0.0025     | 0.0075      | 0.0025      | 0.0042 |

Figure 4: Testing error rates for the synthetic experiments.

|         | 1 vs 2     | 1 vs 3     | 1 vs 4     | 2 vs 3     | 2 vs 4     | 3 vs 4     | total      |
|---------|------------|------------|------------|------------|------------|------------|------------|
| Linear  | 0.0509     | 0.0879     | 0.1381     | 0.0653     | 0.1755     | 0.0941     | 0.1025     |
| TFIDF   | 0.0428     | 0.0891     | 0.1623     | 0.0486     | 0.1910     | 0.1096     | 0.1059     |
| Sqrt    | 0.0363     | **0.0667** | 0.0996     | **0.0456** | 0.1153     | 0.0674     | 0.0711     |
| Poly    | 0.0499     | 0.0861     | 0.1389     | 0.0599     | 0.1750     | 0.0950     | 0.1009     |
| RBF     | 0.0514     | 0.0836     | 0.1356     | 0.0641     | 0.1755     | 0.0981     | 0.1024     |
| L Mono  | 0.0338     | 0.0739     | 0.0854     | 0.0511     | 0.1060     | 0.0602     | 0.0683     |
| C Mono  | **0.0322** | 0.0776     | **0.0812** | 0.0501     | **0.0973** | **0.0584** | **0.0657** |

Figure 5: Testing error rates for WebKB.

These web pages are then categorized. We will be working with the largest four categories: student, faculty, course, and project. The task is to solve all six pairwise classification problems. In [6, 5] preprocessing the data with a square root was demonstrated to yield good results. We will compare our nonconvex method (L mono), and our convex relaxation (C mono) to a linear SVM with and without the square root, with TFIDF features and also a kernelized SVM with both the polynomial kernel and the RBF kernel. We will follow the setup of [6] by training on three universities and the miscellaneous university set and testing on web pages from the fourth university. We repeated this four fold experiment five times. For each fold, we use a subset of 200 points for training, 200 to cross validate the parameter settings, and all of the fourth university's points for testing.

Our two methods outperform the competition on average as reported in Figure 5. The convex relaxation chooses a step function nearly every time. This outputs a 1 if a word is in the training vector and 0 if it is absent. The nonconvex greedy algorithm does not end up recovering this solution as reliably and seems to get stuck in local minima. This leads to slightly worse performance than the convex version.

### 4.3  Image Histogram Classification

In this experiment, we used the Corel image dataset. In [3], it was shown that monotonic transforms of the form $x^a$ for $0 \leq a \leq 1$ worked well. The Corel image dataset is made up of various categories, each containing 100 images. We chose four categories of animals: 1) eagles, 2) elephants, 3) horses, and 4) tigers. Images were transformed into RGB histograms following the binning strategy of [3, 5]. We ran a series of six pairwise experiments where the data was randomly split into 80 percent training, 10 percent cross validation, and 10 percent testing. These six experiments were repeated 10 times. We compared our two methods to a linear support vector machine, as well as an SVM with RBF and polynomial kernels. We also compared to the set of transforms $x^a$ for $0 \leq a \leq 1$ where we cross validated over $a = \{0, .125, .25, .5, .625, .75, .875, 1\}$. This set includes linear $a = 1$ at one end, a binary threshold $a = 0$ at the other (choosing $0^0 = 0$), and the square root transform in the middle.

The convex relaxation performed best or tied for best on 4 out 6 of the experiments and was the best overall as reported in Figure 6. The nonconvex version also performed well but ended up with a lower accuracy than the cross validated family of $x^a$ transforms. The key to this dataset is that most of the data is very close to zero due to few pixels being in a given bin. Cross validation over $x^a$ most often chose low nonzero $a$ values. Our method had many knots in these extremely low values because that was where the data support was. Plots of our learned functions on these small values can be found in Figure 7(a-f). Solid blue is the mean for the nonconvex algorithm and dashed blue is the standard deviation. Similarly, the convex relaxation is in red.

|         | 1 vs 2 | 1 vs 3 | 1 vs 4 | 2 vs 3 | 2 vs 4 | 3 vs 4 | total |
|---------|--------|--------|--------|--------|--------|--------|-------|
| Linear  | 0.08   | 0.10   | 0.28   | 0.11   | 0.14   | 0.26   | 0.1617 |
| Sqrt    | **0.03** | 0.05 | 0.09   | 0.12   | 0.08   | 0.20   | 0.0950 |
| Poly    | 0.07   | 0.10   | 0.28   | 0.11   | 0.15   | 0.23   | 0.1567 |
| RBF     | 0.06   | 0.08   | 0.22   | 0.10   | 0.13   | 0.23   | 0.1367 |
| $x^a$   | 0.08   | 0.04   | **0.03** | **0.03** | 0.09 | 0.06   | 0.0550 |
| L Mono  | 0.05   | 0.06   | 0.04   | 0.05   | 0.13   | **0.05** | 0.0633 |
| C Mono  | 0.04   | **0.03** | **0.03** | 0.04 | **0.06** | **0.05** | **0.0417** |

Figure 6: Testing error rates on Corel dataset.

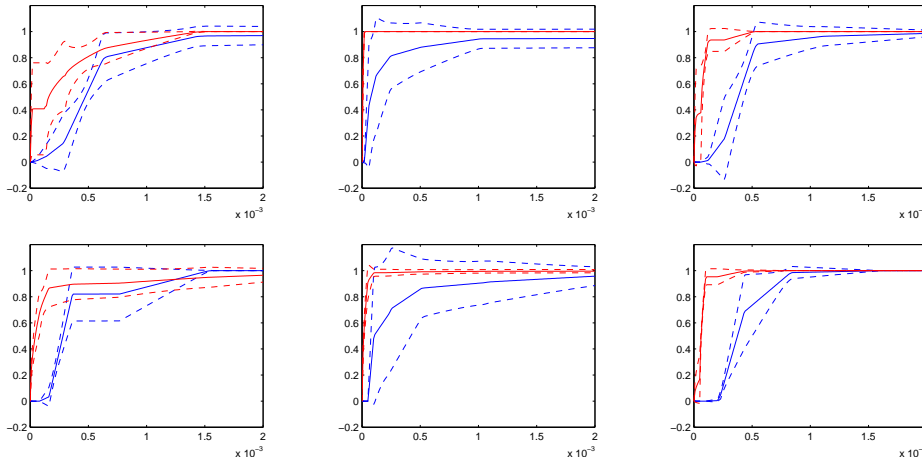

Figure 7: The learned transformation functions for 6 Corel problems.

## 4.4 Gender classification

In this experiment we try to differentiate between images of males and females. We have 1755 labelled images from the FERET dataset processed as in [9]. Each processed image is a 21 by 12 pixel 256 color gray scale image that is rastorized to form training vectors. There are 1044 male images and 711 female images. We randomly split the data into 80 percent training, 10 percent cross validation, and and 10 percent testing. We then compare a linear SVM to our two methods on 5 random splits of the data. The learned monotonic function from L Mono and C Mono are similar to a sigmoid function which indicates that useful saturation and threshold effects where uncovered by our methods. Figure 8a shows examples of training images before and after they have been transformed by our learned function. Figure 8b summarizes the results. Our learned transformation outperforms the linear SVM with the convex relaxation performing best.

## 5 Discussion

A data driven framework was presented for jointly learning monotonic transformations of input data and a discriminative linear classifier. The joint optimization improves classification accuracy and produces interesting transformations that otherwise would require a priori domain knowledge. Two implementations were discussed. The first is a fast greedy algorithm for finding a locally optimal solution. Subsequently, a semidefinite relaxation of the original problem was presented which does not suffer from local minima. The greedy algorithm has similar scaling properties as a support vector machine yet has local minima to contend with. The semidefinite relaxation is more computationally intensive yet ensures a reliable global solution. Nevertheless, both implementations were helpful in synthetic and real experiments including text and image classification and improved over standard support vector machine tools.

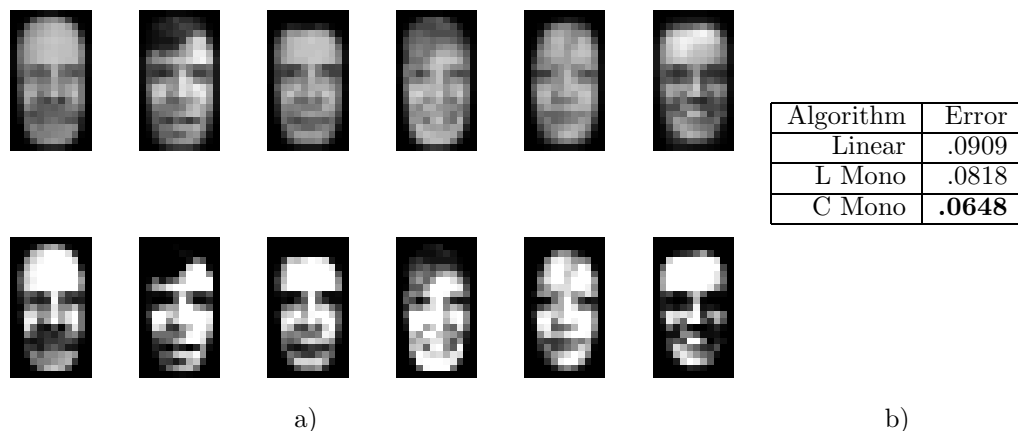

| Algorithm | Error |
|-----------|-------|
| Linear | .0909 |
| L Mono | .0818 |
| C Mono | **.0648** |

a)                                                                    b)

Figure 8: a) Original and transformed gender images. b) Error rates for gender classification.

A natural next step is to explore faster (convex) algorithms that take advantage of the specific structure of the problem. These faster algorithms will help us explore extensions such as learning transformations across multiple tasks. We also hope to explore applications to other domains such as gene expression data to refine the current logarithmic transforms necessary to compensate for well-known saturation effects in expression level measurements. We are also interested in looking at fMRI and audio data where monotonic transformations are useful.

## 6  Acknowledgements

This work was supported in part by NSF Award IIS-0347499 and ONR Award N000140710507.

## References

[1] S. Boyd and L. Vandenberghe. *Convex Optimization*. Cambridge University Press, 2004.

[2] M. Brown, W. Grundy, D. Lin, N. Christianini, C. Sugnet, M. Jr, and D. Haussler. Support vector machine classification of microarray gene expression data, 1999.

[3] O. Chapelle, P. Hafner, and V.N. Vapnik. Support vector machines for histogram-based classification. *Neural Networks, IEEE Transactions on*, 10:1055–1064, 1999.

[4] C. Cortes and V. Vapnik. Support-vector networks. *Machine Learning*, 20(3):273–297, 1995.

[5] M. Hein and O. Bousquet. Hilbertian metrics and positive definite kernels on probability measures. In *Proceedings of Artificial Intelligence and Statistics*, 2005.

[6] T. Jebara, R. Kondor, and A. Howard. Probability product kernels. *Journal of Machine Learning Research*, 5:819–844, 2004.

[7] G. Lanckriet, N. Cristianini, P. Bartlett, L. El Ghaoui, and M. I. Jordan. Learning the kernel matrix with semidefinite programming. *Journal of Machine Learning Research*, 5:27–72, 2004.

[8] J.B. Lasserre. Convergent LMI relaxations for nonconvex quadratic programs. In *Proceedings of 39th IEEE Conference on Decision and Control*, 2000.

[9] B. Moghaddam and M.H. Yang. Sex with support vector machines. In Todd K. Leen, Thomas G. Dietterich, and Volker Tresp, editors, *Advances in Neural Information Processing 13*, pages 960–966. MIT Press, 2000.

[10] T. Robertson, F.T. Wright, and R.L. Dykstra. *Order Restricted Statistical Inference*. Wiley, 1988.
